# A variational mean-field theory for sigmoidal belief networks

**C. Bhattacharyya**
Computer Science and Automation
Indian Institute of Science
Bangalore, India, 560012
*cbchiru@csa.iisc.ernet.in*

**S. Sathiya Keerthi**
Mechanical and Production Engineering
National University of Singapore
*mpessk@guppy.mpe.nus.edu.sg*

## Abstract

A variational derivation of Plefka's mean-field theory is presented. This theory is then applied to sigmoidal belief networks with the aid of further approximations. Empirical evaluation on small scale networks show that the proposed approximations are quite competitive.

## 1 Introduction

Application of mean-field theory to solve the problem of inference in Belief Networks(BNs) is well known [1]. In this paper we will discuss a variational mean-field theory and its application to BNs, sigmoidal BNs in particular.

We present a variational derivation of the mean-field theory, proposed by Plefka[2]. The theory will be developed for a stochastic system, consisting of N binary random variables, $S_i \in \{0,1\}$, described by the energy function $E(\vec{S})$, and the following Boltzmann Gibbs distribution at a temperature $T$:

$$P(\vec{S}) = \frac{e^{-\frac{E(\vec{S})}{T}}}{Z} \quad , \quad Z = \sum_{\vec{S}} e^{-\frac{E(\vec{S})}{T}}.$$

The application of this mean-field method to Boltzmann Machines(BMs) is already done [3]. A large class of BNs are described by the following energy function:

$$E(\vec{S}) = -\sum_{i=1}^{N}\{S_i \ln f(M_i) + (1 - S_i)\ln(1 - f(M_i)\} \quad M_i = \sum_{j=1}^{i-1} w_{ij}S_j + h_i$$

The application of the mean-field theory for such energy functions is not straightforward and further approximations are needed. We propose a new approximation scheme and discuss its utility for sigmoid networks, which is obtained by substituting

$$f(x) = \frac{1}{1 + e^{-x}}$$

in the above energy function. The paper is organized as follows. In section 2 we present a variational derivation of Plefka's mean-field theory. In section 3 the theory is extended to sigmoidal belief networks. In section 4 empirical evaluation is done. Concluding remarks are given in section 5.

## 2  A Variational mean-field theory

Plefka,[2] proposed a mean-field theory in the context of spin glasses. This theory can, in principle, yield arbitrarily close approximation to $\log Z$. In this section we present an alternate derivation from a variational viewpoint, see also [4],[5].

Let $\gamma$ be a real parameter that takes values from 0 to 1. Let us define a $\gamma$ dependent partition and distribution function,

$$Z_\gamma = \sum_{\vec{S}} e^{-\gamma E(\vec{S})/T} \ , \ p_\gamma = \frac{e^{-\gamma E(\vec{S})/T}}{Z_\gamma} \tag{1}$$

Note that $Z_1 = Z$ and $p_1 = p$. Introducing an external real vector, $\vec{\theta}$ let us rewrite (1) as

$$Z_\gamma = \sum_{\vec{S}} \frac{e^{-\gamma \frac{E}{T} + \sum_i \theta_i S_i}}{\widetilde{Z}} e^{-\sum_i \theta_i S_i} \widetilde{Z} \tag{2}$$

where $\widetilde{Z}$ is the partition function associated with the distribution function $\tilde{p}_\gamma$ given by

$$\widetilde{Z} = \sum_{\vec{S}} e^{-\gamma \frac{E}{T} + \sum_i \theta_i S_i} \ , \ \tilde{p}_\gamma = \frac{e^{-\gamma \frac{E}{T} + \sum_i \theta_i S_i}}{\widetilde{Z}} \tag{3}$$

Using Jensen's Inequality, $\langle e^{-x} \rangle \geq e^{-\langle x \rangle}$, we get

$$Z_\gamma = \widetilde{Z} \sum_{\vec{S}} \tilde{p}_\gamma e^{-\sum_i \theta_i S_i} \geq \widetilde{Z} e^{-\sum_i \theta_i u_i} \tag{4}$$

where

$$u_i = \langle S_i \rangle_{\tilde{p}_\gamma} \tag{5}$$

Taking logarithms on both sides of (4) we obtain

$$\log Z_\gamma \geq \log \widetilde{Z} - \sum_i \theta_i u_i \tag{6}$$

The right hand side is defined as a function of $\vec{u}$ and $\gamma$ via the following assumption.

**Invertibility assumption:** For each fixed $\vec{u}$ and $\gamma$, (5) can be solved for $\vec{\theta}$.

If the invertibility assumption holds then we can use $\vec{u}$ as the independent vector (with $\vec{\theta}$ dependent on $\vec{u}$) and rewrite (6) as

$$-\ln Z_\gamma \leq G(\vec{u}, \gamma) \tag{7}$$

where $G$ is as defined in

$$G(\vec{u}, \gamma) = -\ln \widetilde{Z} + \sum_i \theta_i u_i.$$

This then gives a variational feel : treat $\vec{u}$ as an external variable vector and choose it to minimize $G$ for a fixed $\gamma$. The stationarity conditions of the above minimization problem yield

$$\theta_i = \frac{\partial G}{\partial u_i} = 0.$$

At the minimum point we have the equality $G = -\log Z_\gamma$.

It is difficult to invert (5) for $\gamma \neq 0$, thus making it impossible to write an algebraic expression for $G$ for any nonzero $\gamma$. At $\gamma = 0$ the inversion is straightforward and one obtains

$$G(\vec{u}, 0) = \sum_{i=1}^{N} (u_i \ln u_i + (1 - u_i) \ln(1 - u_i)) , \quad \tilde{p}_0 = \prod_i u_i (1 - u_i).$$

A Taylor series approach is then undertaken around $\gamma = 0$ to build an approximation to $G$. Define

$$\widetilde{G}_M = G(\vec{u}, 0) + \sum_k \frac{\gamma^k}{k!} \left. \frac{\partial^k G}{\partial \gamma^k} \right|_{\gamma=0} \tag{8}$$

Then $\widetilde{G}_M$ can be considered as an approximation of $G$. The stationarity conditions are enforced by setting

$$\theta_i = \frac{\partial G}{\partial u_i} \approx \frac{\partial \widetilde{G}_M}{\partial u_i} = 0.$$

In this paper we will restrict ourselves to $M = 2$. To do this we need to evaluate the following derivatives

$$\left. \frac{\partial G}{\partial \gamma} \right|_{\gamma=0} = \langle E \rangle_{\tilde{p}_0} \tag{9}$$

$$\left. \frac{\partial^2 G}{\partial \gamma^2} \right|_{\gamma=0} = -\frac{1}{T^2} \left( \langle (E - \langle E \rangle_{\tilde{p}_0})^2 \rangle_{\tilde{p}_0} - \sum_i \frac{cov^2(E, S_i)}{var(S_i)} \right) \tag{10}$$

where

$$cov(E, S_i) = \langle (E - \langle E \rangle_{\tilde{p}_0})(S_i - u_i) \rangle_{\tilde{p}_0} , \quad var(S_i) = \langle (S_i - u_i)^2 \rangle_{\tilde{p}_0}.$$

For $M = 1$ we have the standard mean-field approach. The expression for $M = 2$ can be identified with the TAP correction. The term (10) yields the TAP term for BM energy function.

## 3  Mean-field approximations for BNs

The method, as developed in the previous section, is not directly useful for BNs because of the intractability of the partial derivatives at $\gamma = 0$. To overcome this problem, we suggest an approximation based on Taylor series expansion. Though in this paper we will be restricting ourselves to sigmoid activation function, this method is applicable to other activation functions also. This method enables calculation of all the necessary terms required for extending Plefka's method for BNs. Since, for BN operation $T$ is fixed to 1, $T$ will be dropped from all equations in the rest of the paper.

Let us define a new energy function

$$\widehat{E}(\beta, \vec{S}, \vec{u}, \vec{w}) = -\sum_{i=1}^{N} \{ S_i \ln f(\widehat{M_i}(\beta)) + (1 - S_i) \ln(1 - f(\widehat{M_i}(\beta))) \} \qquad (11)$$

where $0 \le \beta \le 1$,

$$\widehat{M_i}(\beta) = \sum_{j=1}^{i-1} w_{ij} \beta(S_j - u_j) + \overline{M_i} , \quad \overline{M_i} = \sum_{j=1}^{i-1} w_{ij} u_j + h_i$$

where

$$u_k = \sum_{\vec{S}} S_k p_{\gamma\beta} \ \forall k , \quad p_{\gamma\beta} = \frac{e^{-\gamma \widehat{E} + \sum_i \theta_i S_i}}{\sum_{\vec{s}} e^{-\gamma \widehat{E} + \sum_i \theta_i S_i}} \qquad (12)$$

Since $\beta$ is the important parameter, $\widehat{E}(\beta, \vec{S}, \vec{u}, \vec{w})$ will be referred to as $\widehat{E}(\beta)$ so as to avoid notational clumsiness. We use a Taylor series approximation of $\widehat{E}(\beta)$ with respect to $\beta$. Let us define

$$\widehat{E}_C(\beta) = \widehat{E}(0) + \sum_{k=1}^{C} \frac{\beta^k}{k!} \left. \frac{\partial^k \widehat{E}}{\partial \beta^k} \right|_{\beta=0} . \qquad (13)$$

If $\widehat{E}_C$ approximates $\widehat{E}$, then we can write

$$E = \widehat{E}(1) \approx \widehat{E}_C(1). \qquad (14)$$

Let us now define the following function

$$A(\gamma, \beta, \vec{u}) = -\ln \sum_{\vec{s}} e^{-\gamma \widehat{E} + \sum_i \theta_i S_i} + \sum_i \theta_i u_i \qquad (15)$$

The $\theta_i$ are assumed to be functions of $\vec{u}, \beta, \gamma$, which are obtained by inverting equations (12) By replacing $\widehat{E}$ by $\widehat{E}_C$ in (15) we obtain $A_C$

$$A_C(\gamma, \beta, \vec{u}) = -\ln \sum_{\vec{s}} e^{-\gamma \widehat{E}_C + \sum_i \theta_i S_i} + \sum_i \theta_i u_i \qquad (16)$$

where the definition of $\vec{u}$ is obtained by replacing $\widehat{E}$ by $\widehat{E}_C$. In view of (14) one can consider $A_C$ as an approximation to $A$. This observation suggests an approximation to $G$.

$$G(\gamma, \vec{u}) = A(\gamma, 1, \vec{u}) \approx A_C(\gamma, 1, \vec{u}) \qquad (17)$$

The required terms needed in the Taylor expansion of $G$ in $\gamma$ can be approximated by

$$G(0, \vec{u}) = A(0, 1, \vec{u}) = A_C(0, 1, \vec{u})$$

$$\left. \frac{\partial^k G}{\partial \gamma^k} \right|_{\gamma=0} = \left. \frac{\partial^k A}{\partial \gamma^k} \right|_{\gamma=0, \beta=1} \approx \left. \frac{\partial^k A_C}{\partial \gamma^k} \right|_{\gamma=0, \beta=1}$$

The biggest advantage in working with $A_C$ rather than $G$ is that the partial derivatives of $A_C$ with respect to $\gamma$ at $\gamma = 0$ and $\beta = 1$ can be expressed as functions of $\vec{u}$. We define

$$\widehat{G}_{MC}(\vec{u}, \gamma) = A_C(0, 1, \vec{u}) + \sum_{k=1}^{M} \frac{\gamma^k}{k!} \left. \frac{\partial^k A_C}{\partial \gamma^k} \right|_{\gamma=0, \beta=1} \qquad (18)$$

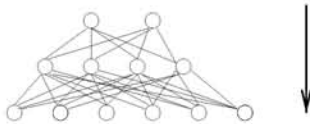

Figure 1: Three layer BN ($2 \times 4 \times 6$) with top down propagation of beliefs. The activation function was chosen to be sigmoid.

In light of the above discussion one can consider $\widetilde{G}_M \approx \widehat{G}_{MC}$; hence the mean-field equations can be stated as

$$\theta_i = \frac{\partial G}{\partial u_i} \approx \frac{\partial \widetilde{G}_M}{\partial u_i} \approx \frac{\partial \widehat{G}_{MC}}{\partial u_i} = 0 \tag{19}$$

In this paper we will restrict ourselves to $M = 2$. The relevant objective functions for a general $C$ is given by

$$\widehat{G}_{1C} = \sum_i u_i \log u_i + (1 - u_i) \log(1 - u_i) + \langle E_C \rangle_{\bar{p}_0} \tag{20}$$

$$\widehat{G}_{2C} = \widehat{G}_{1C} - \frac{1}{2} \left( \langle (E_C - \langle E_C \rangle_{\bar{p}_0})^2 \rangle - \sum_i \frac{cov^2(E_C, S_i)}{var(S_i)} \right) \tag{21}$$

All these objective functions can be expressed as a function of $\vec{u}$.

## 4   Experimental results

To test the approximation schemes developed in the previous schemes, numerical experiments were conducted. Saul et al.[1] pioneered the application of mean-field theory to BNs. We will refer to their method as the SJJ approach. We compare our schemes with the SJJ approach.

Small Networks were chosen so that $\ln Z$ can be computed by exact enumeration for evaluation purposes. For all the experiments the network topology was fixed to the one shown in figure 1. This choice of the network enables us to compare the results with those of [1]. To compare the performance of our methods with their method we repeated the experiment conducted by them for sigmoid BNs. Ten thousand networks were generated by randomly choosing weight values in $[-1, 1]$. The bottom layer units, or the visible units of each network were instantiated to zero. The likelihood, $\ln Z$, was computed by exact enumeration of all the states in the higher two layers. The approximate value of $-\ln Z$ was computed by $\widehat{G}_{MC}$; $\vec{u}$ was computed by solving the fixed point equations obtained from (19). The goodness of approximation scheme was tested by the following measure

$$\mathcal{E} = -\frac{\widehat{G}_{MC}}{\ln Z} - 1 \tag{22}$$

For a proper comparison we also implemented the SJJ method. The goodness of approximation for the SJJ scheme is evaluated by substituting $\widehat{G}_{MC}$, in (22) by $L_{sapprox}$, for specific formula see [1]. The results are presented in the form of histograms in Figure 2. We also repeated the experiment with weights and

|  | $\langle \mathcal{E} \rangle$<br>small weights $[-1, 1]$ | $\langle \mathcal{E} \rangle$<br>large weights $[-5, 5]$ |
|---|---|---|
| $\widehat{G}_{11}$ | -0.0404 | -0.0440 |
| $\widehat{G}_{12}$ | 0.0155 | 0.0231 |
| $\widehat{G}_{22}$ | 0.0029 | -0.0456 |
| $SJJ$ | 0.0157 | 0.0962 |

Table 1: Mean of $\mathcal{E}$ for randomly generated sigmoid networks, in different weight ranges.

biases taking values between -5 and 5, the results are again presented in the form of histograms in Figure 3. The findings are summarized in the form of means tabulated in Table 1.

For small weights $\widehat{G}_{12}$ and the SJJ approach show close results, which was expected. But the improvement achieved by the $\widehat{G}_{22}$ scheme is remarkable; it gave a mean value of 0.0029 which compares substantially well against the mean value of 0.01139 reported in [6]. The improvement in [6] was achieved by using mixture distribution which requires introduction of extra variational variables; more than 100 extra variational variables are needed for a 5 component mixture. This results in substantial increase in the computation costs. On the other hand the extra computational cost for $\widehat{G}_{22}$ over $\widehat{G}_{12}$ is marginal. This makes the $\widehat{G}_{22}$ scheme computationally attractive over the mixture distribution.

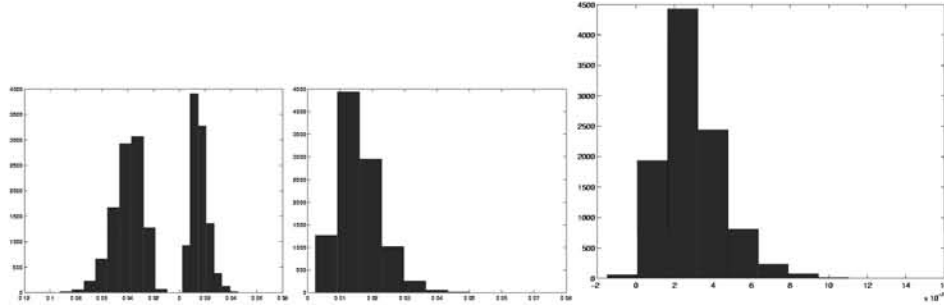

Figure 2: Histograms for $\widehat{G}_{1C}$ and SJJ scheme for weights taking values in $[-1, 1]$, for sigmoid networks. The plot on the left show histograms for $\mathcal{E}$ for the schemes $\widehat{G}_{11}$ and $\widehat{G}_{12}$ They did not have any overlaps; $\widehat{G}_{11}$, gives a mean of -0.040 while $\widehat{G}_{12}$ gives a mean of 0.0155. The middle plot shows the histogram for the SJJ scheme, mean is given by 0.0157.The plot at the extreme right is for the scheme $\widehat{G}_{22}$, having a mean of 0.0029

Of the three schemes $\widehat{G}_{12}$ is the most robust and also yields reasonably accurate results. It is outperformed only by $\widehat{G}_{22}$ in the case of sigmoid networks with low weights. Empirical evidence thus suggests that the choice of a scheme is not straightforward and depends on the activation function and also parameter values.

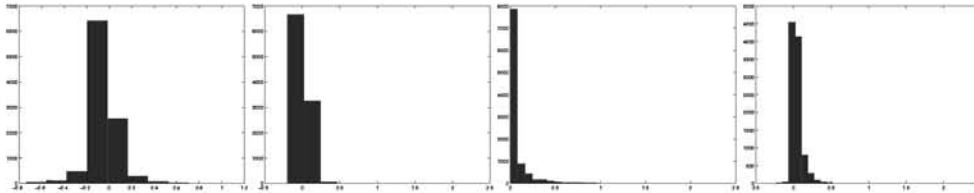

Figure 3: Histograms for the $\widehat{G}_{1C}$ and SJJ schemes for weights taking values in $[-5, 5]$ for sigmoid networks. The leftmost histogram shows $\mathcal{E}$ for $\widehat{G}_{11}$ scheme having a mean of $-0.0440$, second from left is for $\widehat{G}_{12}$ scheme having a mean of $0.0231$, and second from right is for SJJ scheme, having a mean of $0.0962$. The scheme $\widehat{G}_{22}$ is at the extreme right with mean $-0.0456$.

## 5   Discussion

Application of Plefka's theory to BNs is not straightforward. It requires computation of some averages which are not tractable. We presented a scheme in which the BN energy function is approximated by a Taylor series, which gives a tractable approximation to the terms required for Plefka's method. Various approximation schemes depending on the degree of the Taylor series expansion are derived. Unlike the approach in [1], the schemes discussed here are simpler as they do not introduce extra variational variables. Empirical evaluation on small scale networks shows that the quality of approximations is quite good. For a more detailed discussion of these points see [7].

## References

[1] Saul, L. K. and Jaakkola, T. and Jordan, M. I.(1996), Mean field theory for sigmoid belief networks, *Journal of Artificial Intelligence Research,4*

[2] Plefka, T. (1982), Convergence condition of the TAP equation for the Infinite-ranged Ising glass model,*J. Phys. A: Math. Gen.,15*

[3] Kappen, H. J and Rodriguez, F. B(1998), Boltzmann machine learning using mean field theory and linear response correction, *Advances in Neural Information Processing Systems 10*, (eds.) M. I. Jordan and M. J. Kearns and S. A. Solla, MIT press

[4] Georges, A. and Yedidia, J. S.(1991), How to expand around mean-field theory using high temperature expansions,*J. Phys. A: Math. Gen., 24*

[5] Bhattacharyya, C. and Keerthi, S. S.(2000), Information geometry and Plefka's mean-field theory, *J. Phys. A: Math. Gen.,33*

[6] Bishop, M. C. and Lawrence, N. and Jaakkola, T. and Jordan, M. I.(1997), Approximating Posterior Distributions in Belief Networks using Mixtures, *Advances in Neural Information Processing Systems 10*, (eds.) Jordan, M. I. and Kearns, M. J. and Solla, S., MIT press

[7] Bhattacharyya, C. and Keerthi, S. S. (1999), Mean field theory for a special class of belief networks, accepted in *Journal of Artificial Intelligence Research*
